# Variational Inference for Crowdsourcing

**Qiang Liu**
ICS, UC Irvine
qliu1@ics.uci.edu

**Jian Peng**
TTI-C & CSAIL, MIT
jpeng@csail.mit.edu

**Alexander Ihler**
ICS, UC Irvine
ihler@ics.uci.edu

## Abstract

Crowdsourcing has become a popular paradigm for labeling large datasets. However, it has given rise to the computational task of aggregating the crowdsourced labels provided by a collection of unreliable annotators. We approach this problem by transforming it into a standard inference problem in graphical models, and applying approximate variational methods, including belief propagation (BP) and mean field (MF). We show that our BP algorithm generalizes both majority voting and a recent algorithm by Karger et al. [1], while our MF method is closely related to a commonly used EM algorithm. In both cases, we find that the performance of the algorithms critically depends on the choice of a prior distribution on the workers' reliability; by choosing the prior properly, both BP and MF (and EM) perform surprisingly well on both simulated and real-world datasets, competitive with state-of-the-art algorithms based on more complicated modeling assumptions.

## 1 Introduction

Crowdsourcing has become an efficient and inexpensive way to label large datasets in many application domains, including computer vision and natural language processing. Resources such as Amazon Mechanical Turk provide markets where the requestors can post tasks known as HITs (Human Intelligence Tasks) and collect large numbers of labels from hundreds of online workers (or annotators) in a short time and with relatively low cost.

A major problem of crowdsoucing is that the qualities of the labels are often unreliable and diverse, mainly since it is difficult to monitor the performance of a large collection of workers. In the extreme, there may exist "spammers", who submit random answers rather than good-faith attempts to label, or even "adversaries", who may deliberately give wrong answers, either due to malice or to a misinterpretation of the task. A common strategy to improve reliability is to add redundancy, such as assigning each task to multiple workers, and aggregate the workers' labels. The baseline *majority voting* heuristic, which simply assigns the label returned by the majority of the workers, is known to be error-prone, because it counts all the annotators equally. In general, efficient aggregation methods should take into account the differences in the workers' labeling abilities.

A principled way to address this problem is to build generative probabilistic models for the annotation processes, and assign labels using standard inference tools. A line of early work builds simple models characterizing the annotators using confusion matrices, and infers the labels using the EM algorithm [e.g., 2, 3, 4]. Recently however, significant efforts have been made to improve performance by incorporating more complicated generative models [e.g., 5, 6, 7, 8, 9]. However, EM is widely criticized for having local optimality issues [e.g., 1]; this raises a potential tradeoff between more dedicated *exploitation* of the simpler models, either by introducing new inference tools or fixing local optimality issues in EM, and the *exploration* of larger model space, usually with increased computational cost and possibly the risk of over-fitting.

On the other hand, variational approaches, including the popular *belief propagation* (BP) and *mean field* (MF) methods, provide powerful inference tools for probabilistic graphical models [10, 11].

These algorithms are efficient, and often have provably strong local optimality properties or even globally optimal guarantees [e.g., 12]. To our knowledge, no previous attempts have taken advantage of variational tools for the crowdsourcing problem. A closely related approach is a message-passing-style algorithm in Karger et al. [1] (referred to as KOS in the sequel), which the authors asserted to be motivated by but not equivalent to standard belief propagation. KOS was shown to have strong theoretical guarantees on (locally tree-like) random assignment graphs, but does not have an obvious interpretation as a standard inference method on a generative probabilistic model. As one consequence, the lack of a generative model interpretation makes it difficult to either extend KOS to more complicated models or adapt it to improve its performance on real-world datasets.

**Contribution.** In this work, we approach the crowdsourcing problems using tools and concepts from variational inference methods for graphical models. First, we present a belief-propagation-based method, which we show includes both KOS and majority voting as special cases, in which particular prior distributions are assumed on the workers' abilities. However, unlike KOS our method is derived using generative principles, and can be easily extended to more complicated models. On the other side, we propose a mean field method which we show closely connects to, and provides an important perspective on, EM. For both our BP and MF algorithms (and consequently for EM as well), we show that performance can be significantly improved by using more carefully chosen priors. We test our algorithms on both simulated and real-world datasets, and show that both BP and MF (or EM), with carefully chosen priors, is able to perform competitively with state-of-the-art algorithms that are based on far more complicated models.

## 2 Background

Assume there are $M$ workers and $N$ tasks with binary labels $\{\pm 1\}$. Denote by $z_i \in \{\pm 1\}$, $i \in [N]$ the true label of task $i$, where $[N]$ represents the set of first $N$ integers; $\mathcal{N}_j$ is the set of tasks labeled by worker $j$, and $\mathcal{M}_i$ the workers labeling task $i$. The task assignment scheme can be represented by a bipartite graph where an edge $(i, j)$ denotes that the task $i$ is labeled by the worker $j$. The labeling results form a matrix $L \in \{0, \pm 1\}^{N \times M}$, where $L_{ij} \in \{\pm 1\}$ denotes the answer if worker $j$ labels task $i$, and $L_{ij} = 0$ if otherwise. The goal is to find an optimal estimator $\hat{z}$ of the true labels $z$ given the observation $L$, minimizing the average bit-wise error rate $\frac{1}{N} \sum_{i \in [N]} \text{prob}[\hat{z}_i \neq z_i]$.

We assume that all the tasks have the same level of difficulty, but that workers may have different predictive abilities. Following Karger et al. [1], we initially assume that the ability of worker $j$ is measured by a single parameter $q_j$, which corresponds to their probability of correctness: $q_j = \text{prob}[L_{ij} = z_i]$. More generally, the workers' abilities can be measured by a confusion matrix, to which our method can be easily extended (see Section 3.1.2).

The values of $q_j$ reflect the abilities of the workers: $q_j \approx 1$ correspond to *experts* that provide reliable answers; $q_j \approx 1/2$ denote *spammers* that give random labels independent of the questions; and $q_j < 1/2$ denote *adversaries* that tend to provide opposite answers. Conceptually, the spammers and adversaries should be treated differently: the spammers provide no useful information and only degrade the results, while the adversaries actually carry useful information, and can be exploited to improve the results if the algorithm can identify them and flip their labels. We assume the $q_j$ of all workers are drawn independently from a common prior $p(q_j|\theta)$, where $\theta$ are the hyper-parameters. To avoid the cases when adversaries and/or spammers overwhelm the system, it is reasonable to require that $\mathbb{E}[q_j|\theta] > 1/2$. Typical priors include the Beta prior $p(q_j|\theta) \propto q_j^{\alpha-1}(1-q_j)^{\beta-1}$ and discrete priors, e.g., the *spammer-hammer* model, where $q_j \approx 0.5$ or $q_j \approx 1$ with equal probability.

*Majority Voting.* The majority voting (MV) method aggregates the workers' labels by

$$\hat{z}_i^{majority} = \text{sign}[\sum_{j \in \mathcal{M}_i} L_{ij}].$$

The limitation of MV is that it weights all the workers equally, and performs poorly when the qualities of the workers are diverse, especially when adversarial workers exist.

*Expectation Maximization.* Weighting the workers properly requires estimating their abilities $q_j$, usually via a maximum *a posteriori* estimator, $\hat{q} = \arg\max \log p(q|L, \theta) = \log \sum_z p(q, z|L, \theta)$. This is commonly solved using an EM algorithm treating the $z$ as hidden variables, [e.g., 2, 3, 4]. Assuming a $\text{Beta}(\alpha, \beta)$ prior on $q_j$, EM is formulated as

$$\text{E-step:} \quad \mu_i(z_i) \propto \prod_{j \in \mathcal{M}_i} \hat{q}_j^{\delta_{ij}} (1 - \hat{q}_j)^{1 - \delta_{ij}}, \qquad \text{M-step:} \quad \hat{q}_j = \frac{\sum_{i \in \mathcal{N}_j} \mu_i(L_{ij}) + \alpha - 1}{|\mathcal{N}_j| + \alpha + \beta - 2}, \quad (1)$$

where $\delta_{ij} = \mathbb{I}[L_{ij} = z_i]$; the $\hat{z}_i$ is then estimated via $\hat{z}_i = \arg\max_{z_i} \mu_i(z_i)$. Many approaches have been proposed to improve this simple EM approach, mainly by building more complicated models.

*Message Passing.* A rather different algorithm in a message-passing style is proposed by Karger, Oh and Shah [1] (referred to as KOS in the sequel). Let $x_{i \to j}$ and $y_{j \to i}$ be *real-valued* messages from tasks to workers and from workers to tasks, respectively. Initializing $y_{j \to i}^0$ randomly from $\mathrm{Normal}(1, 1)$ or deterministically by $y_{j \to i}^0 = 1$, KOS updates the messages at $t$-th iteration via

$$x_{i \to j}^{t+1} = \sum_{j' \in \mathcal{M}_{i \setminus j}} L_{ij'} y_{j' \to i}^t, \qquad y_{j \to i}^{t+1} = \sum_{i' \in \mathcal{N}_{j \setminus i}} L_{i'j} x_{i' \to j}^{t+1}, \qquad (2)$$

and the labels are estimated via $\hat{s}_i^t = \mathrm{sign}[\hat{x}_i^t]$, where $\hat{x}_i^t = \sum_{j \in \mathcal{M}_i} L_{ij} y_{j \to i}^t$. Note that the 0th iteration of KOS reduces to majority voting when initialized with $y_{j \to i}^0 = 1$. KOS has surprisingly nice theoretical properties on locally tree-like assignment graphs: its error rate is shown to scale in the same manner as an oracle lower bound that assumes the true $q_j$ are known. Unfortunately, KOS is not derived using a generative model approach under either Bayesian or maximum likelihood principles, and hence is difficult to extend to more general cases, such as more sophisticated worker-error models (Section 3.1.2) or other features and side information (see appendix). Given that the assumptions made in Karger et al. [1] are restrictive in practice, it is unclear whether the theoretical performance guarantees of KOS hold in real-world datasets. Additionally, an interesting phase transition phenomenon was observed in Karger et al. [1] – the performance of KOS was shown to degenerate, sometimes performing even worse than majority voting when the degrees of the assignment graph (corresponding to the number of annotators per task) are small.

## 3 Crowdsourcing as Inference in a Graphical Model

We present our main framework in this section, transforming the labeling aggregation problem into a standard inference problem on a graphical model, and proposing a set of efficient variational methods, including a belief propagation method that includes KOS and majority voting as special cases, and a mean field method, which connects closely to the commonly used EM approach.

To start, the joint posterior distribution of workers' abilities $q = \{q_j : j \in [M]\}$ and the true labels $z = \{z_i : i \in [N]\}$ conditional on the observed labels $L$ and hyper-parameter $\theta$ is

$$p(z, q | L, \theta) \propto \prod_{j \in [M]} p(q_j | \theta) \prod_{i \in \mathcal{N}_j} p(L_{ij} | z_i, q_j) = \prod_{j \in [M]} p(q_j | \theta) q_j^{c_j} (1 - q_j)^{\gamma_j - c_j},$$

where $\gamma_j = |\mathcal{N}_j|$ is the number of predictions made by worker $j$ and $c_j := \sum_{i \in \mathcal{N}_j} \mathbb{I}[L_{ij} = z_i]$ is the number of $j$'s predictions that are correct. By standard Bayesian arguments, one can show that the optimal estimator of $z$ to minimize the bit-wise error rate is given by

$$\hat{z}_i = \arg\max_{z_i} p(z_i | L, \theta) \qquad \text{where} \qquad p(z_i | L, \theta) = \sum_{z_{[N] \setminus i}} \int_q p(z, q | L, \theta) dq. \qquad (3)$$

Note that the EM algorithm (1), which *maximizes* rather than *marginalizes* $q_j$, is not equivalent to the Bayesian estimator (3), and hence is expected to be suboptimal in terms of error rate. However, calculating the marginal $p(z_i | L, \theta)$ in (3) requires integrating all $q$ and summing over all the other $z_i$, a challenging computational task. In this work we use belief propagation and mean field to address this problem, and highlight their connections to KOS, majority voting and EM.

### 3.1 Belief Propagation, KOS and Majority Voting

It is difficult to directly apply belief propagation to the joint distribution $p(z, q | L, \theta)$, since it is a mixed distribution of discrete variables $z$ and continuous variables $q$. We bypass this issue by directly integrating over $q_j$, yielding a marginal posterior distribution over the discrete variables $z$,

$$p(z | L, \theta) = \int p(z, q | L, \theta) dq = \prod_{j \in [M]} \int_0^1 p(q_j | \theta) q_j^{c_j} (1 - q_j)^{\gamma_j - c_j} \overset{def}{=} \prod_{j \in [M]} \psi_j(z_{\mathcal{N}_j}), \quad (4)$$

where $\psi_j(z_{\mathcal{N}_j})$ is the local factor contributed by worker $j$ due to eliminating $q_j$, which couples all the tasks $z_{\mathcal{N}_j}$ labeled by $j$; here we suppress the dependency of $\psi_j$ on $\theta$ and $L$ for notational simplicity. A key perspective is that we can treat $p(z|L, \theta)$ as a discrete Markov random field, and re-interpret the bipartite assignment graph as a *factor graph* [13], with the tasks mapping to variable nodes and workers to factor nodes. This interpretation motivates us to use a standard sum-product belief propagation method, approximating $p(z_i|L, \theta)$ with "beliefs" $b_i(z_i)$ using messages $m_{i \to j}$ and $m_{j \to i}$ between the variable nodes (tasks) and factor nodes (workers),

$$\text{From tasks to workers:} \qquad m_{i \to j}^{t+1}(z_i) \propto \prod_{j' \in \mathcal{M}_{i/j}} m_{j' \to i}^t(z_i), \qquad (5)$$

$$\text{From workers to tasks:} \qquad m_{j \to i}^{t+1}(z_i) \propto \sum_{z_{\mathcal{N}_{j/i}}} \psi_j(z_{\mathcal{N}_j}) \prod_{i' \in \mathcal{N}_j} m_{i' \to j}^{t+1}(z_{i'}), \qquad (6)$$

$$\text{Calculating the beliefs:} \qquad b_i^{t+1}(z_i) \propto \prod_{j \in \mathcal{M}_i} m_{j \to i}^{t+1}(z_i). \qquad (7)$$

At the end of T iterations, the labels are estimated via $\hat{z}_i^t = \arg\max_{z_i} b_i^t(z_i)$. One immediate difference between BP (5)-(7) and the KOS message passing (2) is that the messages and beliefs in (5)-(7) are probability tables on $z_i$, i.e., $m_{i \to j} = [m_{i \to j}(+1), m_{i \to j}(-1)]$, while the messages in (2) are real values. For binary labels, we will connect the two by rewriting the updates (5)-(7) in terms of their (real-valued) log-odds, a standard transformation used in error-correcting codes.

The BP updates above appear computationally challenging, since step (6) requires eliminating a high-order potential $\psi(z_{\mathcal{N}_j})$, costing $O(2^{\gamma_j})$ in general. However, note that $\psi(z_{\mathcal{N}_j})$ in (4) depends on $z_{\mathcal{N}_j}$ only through $c_j$, so that (with a slight abuse of notation) it can be rewritten as $\psi(c_j, \gamma_j)$. This structure enables us to rewrite the BP updates in a more efficient form (in terms of the log-odds):

**Theorem 3.1.**

$$\text{Let} \quad \hat{x}_i = \log \frac{b_i(+1)}{b_i(-1)}, \quad x_{i \to j} = \log \frac{m_{i \to j}(+1)}{m_{i \to j}(-1)}, \quad \text{and} \quad y_{j \to i} = L_{ij} \log \frac{m_{j \to i}(+1)}{m_{i \to j}(-1)}.$$

*Then, sum-product BP* (5)-(7) *can be expressed as*

$$x_{i \to j}^{t+1} = \sum_{j' \in \mathcal{M}_{i \setminus j}} L_{ij} y_{j' \to i}^t, \qquad y_{j \to i}^{t+1} = \log \frac{\sum_{k=0}^{\gamma_j - 1} \psi(k+1, \gamma_j) \, e_k^{t+1}}{\sum_{k=0}^{\gamma_j - 1} \psi(k, \gamma_j) \, e_k^{t+1}}, \qquad (8)$$

*and* $\hat{x}_i^{t+1} = \sum_{j \in \mathcal{M}_i} L_{ij} y_{j \to i}^{t+1}$, *where the terms* $e_k$ *for* $k = 0, \ldots, N_j - 1$, *are the elementary symmetric polynomials in variables* $\{\exp(L_{i'j} x_{i' \to j})\}_{i' \in \mathcal{N}_{j \setminus i}}$, *that is,* $e_k = \sum_{s:\, |s|=k} \prod_{i' \in s} \exp(L_{i'j} x_{i' \to j})$. *In the end, the true labels are decoded as* $\hat{z}_i^t = \text{sign}[\hat{x}_i^t]$.

The terms $e_k$ can be efficiently calculated by divide & conquer and the fast Fourier transform in $O(\gamma_j (\log \gamma_j)^2)$ time (see appendix), making (8) much more efficient than (6) initially appears.

Similar to sum-product, one can also derive a max-product BP to find the joint maximum *a posteriori* configuration, $\hat{z} = \arg\max_z p(z|L, \theta)$, which minimizes the block-wise error rate $\text{prob}[\exists i : z_i \neq \hat{z}_i]$ instead of the bit-wise error rate. Max-product BP can be organized similarly to (8), with the slightly lower computational cost of $O(\gamma_j \log \gamma_j)$; see appendix for details and Tarlow et al. [14] for a general discussion on efficient max-product BP with structured high-order potentials. In this work, we focus on sum-product since the bit-wise error rate is more commonly used in practice.

### 3.1.1 The Choice of Algorithmic Priors and connection to KOS and Majority Voting

Before further discussion, we should be careful to distinguish between the prior on $q_j$ used in our algorithm (the *algorithmic prior*) and, assuming the model is correct, the true distribution of the $q_j$ in the data generating process (the *data prior*); the algorithmic and data priors often do not match. In this section, we discuss the form of $\psi(c_j, \gamma_j)$ for different choices of algorithmic priors, and in particular show that KOS and majority voting can be treated as special cases of our belief propagation (8) with the most "uninformative" and most "informative" algorithmic priors, respectively. For more general priors that may not yield a closed form for $\psi(c_j, \gamma_j)$, one can calculate $\psi(c_j, \gamma_j)$ by numerical integration and store them in a $(\gamma + 1) \times \gamma$ table for later use, where $\gamma = \max_{j \in [M]} \gamma_j$.

**Beta Priors.** If $p(q_j|\theta) \propto q_j^{\alpha-1}(1-q_j)^{\beta-1}$, we have $\psi(c_j, \gamma_j) \propto B(\alpha + c_j, \beta + \gamma_j - c_j)$, where $B(\cdot, \cdot)$ is the Beta function. Note that $\psi(c_j, \gamma_j)$ in this case equals (up to a constant) the likelihood of a Beta-binomial distribution.

**Discrete Priors.** If $p(q_j|\theta)$ has non-zero probability mass on only finite points, that is, $\mathrm{prob}(q_j = \tilde{q}_k) = p_k$, $k \in [K]$, where $0 \le \tilde{q}_k \le 1$, $0 \le p_k \le 1$ and $\sum_k p_k = 1$, then we have $\psi(c_j, \gamma_j) = \sum_k p_k \tilde{q}_k^{c_j}(1 - \tilde{q}_k)^{\gamma_j - c_j}$. One can show that $\log \psi(c_j, \gamma_j)$ in this case is a log-sum-exp function.

**Haldane Prior.** The Haldane prior [15] is a special discrete prior that equals either 0 or 1 with equal probability, that is, $\mathrm{prob}[q_j = 0] = \mathrm{prob}[q_j = 1] = 1/2$. One can show that in this case we have $\psi(0, \gamma_j) = \psi(\gamma_j, \gamma_j) = 1$ and $\psi(c_j, \gamma_j) = 0$ otherwise.

**Claim 3.2.** *The BP update in* (8) *with Haldane prior is equivalent to KOS update in* (2).

*Proof.* Just substitute the $\psi(c_j, \gamma_j)$ of Haldane prior shown above into the BP update (8). $\qquad\square$

The Haldane prior can also be treated as a $\mathrm{Beta}(\epsilon, \epsilon)$ prior with $\epsilon \to 0^+$, or equivalently an improper prior $p(q_j) \propto q_j^{-1}(1 - q_j)^{-1}$, whose normalization constant is infinite. One can show that the Haldane prior is equivalent to putting a flat prior on the log-odds $\log[q_j/(1 - q_j)]$; also, it has the largest variance (and hence is "most uninformative") among all the possible distributions of $q_j$. Therefore, although appearing to be extremely dichotomous, it is well known in Bayesian statistics as an *uninformative prior* of binomial distributions. Other choices of objective priors include the uniform prior $\mathrm{Beta}(1, 1)$ and Jeffery's prior $\mathrm{Beta}(1/2, 1/2)$ [16], but these do not yield the same simple linear message passing form as the Haldane prior.

Unfortunately, the use of Haldane prior in our problem suffers an important symmetry breaking issue: if the prior is symmetric, i.e., $p(q_j|\theta) = p(1 - q_j|\theta)$, the true marginal posterior distribution of $z_j$ is also symmetric, i.e., $p(z_j|L, \theta) = [1/2; 1/2]$, because jointly flipping the sign of any configuration does not change its likelihood. This makes it impossible to break the ties when decoding $z_j$. Indeed, it is not hard to observe that $x_{i \to j} = y_{j \to i} = 0$ (corresponding to symmetric probabilities) is a fixed point of the KOS update (2). The mechanism of KOS for breaking the symmetry seems to rely solely on initializing to points that bias towards majority voting, and the hope that the symmetric distribution is an unstable fixed point. In experiments, we find that the use of symmetric priors usually leads to degraded performance when the degree of the assignment graph is low, corresponding to the phase transition phenomenon discussed in Karger et al. [1]. This suggests that it is beneficial to use asymmetric priors with $\mathbb{E}[q_j|\theta] > 1/2$, to incorporate the prior knowledge that the majority of workers are non-adversarial. Interestingly, it turns out that majority voting uses such an asymmetric prior, but unfortunately corresponding to another unrealistic extreme.

**Deterministic Priors.** A deterministic prior is a special discrete distribution that equals a single point deterministically, i.e., $\mathrm{prob}[q_j = \tilde{q}|\theta] = 1$, where $0 \le \tilde{q} \le 1$. One can show that $\log \psi$ in this case is a linear function, that is, $\log \psi(c_j, \gamma_j) = c_j \mathrm{logit}(\tilde{q}) + const$.

**Claim 3.3.** *The BP update* (8) *with deterministic priors satisfying $\tilde{q} > 1/2$ terminates at the first iteration and finds the same solution as majority voting.*

*Proof.* Just note that $\log \psi(c_j, \gamma_j) = c_j \mathrm{logit}(\tilde{q}) + const$, and $\mathrm{logit}(\tilde{q}) > 0$ in this case. $\qquad\square$

The deterministic priors above have the opposite properties to the Haldane prior: they can be also treated as $\mathrm{Beta}(\alpha, \beta)$ priors, but with $\alpha \to +\infty$ and $\alpha > \beta$; these priors have the *smallest* variance (equal to zero) among all the possible $q_j$ priors.

In this work, we propose to use priors that balance between KOS and majority voting. One reasonable choice is $\mathrm{Beta}(\alpha, 1)$ prior with $\alpha > 1$ [17]. In experiments, we find that a typical choice of $\mathrm{Beta}(2, 1)$ performs surprisingly well even when it is far from the true prior.

### 3.1.2 The Two-Coin Models and Further Extensions

We previously assumed that workers' abilities are parametrized by a single parameter $q_j$. This is likely to be restrictive in practice, since the error rate may depend on the true label value: false positive and false negative rates are often not equal. Here we consider the more general case, where the ability of worker $j$ is specified by two parameters, the *sensitivity* $s_j$ and *specificity* $t_j$ [2, 4],

$$s_j = \mathrm{prob}[L_{ij} = +1|z_i = +1], \qquad t_j = \mathrm{prob}[L_{ij} = -1|z_i = -1].$$

A typical prior on $s_j$ and $t_j$ are two independent Beta distributions. One can show that $\psi(z_{\mathcal{N}_j})$ in this case equals a product of two Beta functions, and depends on $z_{\mathcal{N}_j}$ only through two integers, the true positive and true negative counts. An efficient BP algorithm similar to (8) can be derived for the general case, by exploiting the special structure of $\psi(z_{\mathcal{N}_j})$. See the Appendix for details.

One may also try to derive a two-coin version of KOS, by assigning two independent Haldane priors on $s_j$ and $t_j$; it turns out that the extended version is exactly the same as the standard KOS in (2). In this sense, the Haldane prior is too restrictive for the more general case. Several further extensions of the BP algorithm that are not obvious for KOS, for example the case when known features of the tasks or other side information are available, are discussed in the appendix due to space limitations.

## 3.2 Mean Field Method and Connection of EM

We next present a mean field method for computing the marginal $p(z_i|L,\theta)$ in (3), and show its close connection to EM. In contrast to the derivation of BP, here we directly work on the mixed joint posterior $p(z,q|L,\theta)$. Let us approximate $p(z,q|L,\theta)$ with a fully factorized distribution $b(z,q) = \prod_{i\in[N]} \mu_i(z_i) \prod_{j\in[M]} \nu_j(q_j)$. The best $b(z,q)$ should minimize the KL divergence,

$$\mathrm{KL}[b(z,q) \,||\, p(z,q|L,\theta)] = -\mathbb{E}_b[\log p(z,q|L,\theta)] - \sum_{i\in[N]} H(\mu_i) - \sum_{j\in[M]} H(\nu_j).$$

where $\mathbb{E}_b[\cdot]$ denotes the expectation w.r.t. $b(z,q)$, and $H(\cdot)$ the entropy functional. Assuming the algorithmic prior of $\mathrm{Beta}(\alpha,\beta)$, one crucial property of the KL objective in this case is that the optimal $\{\nu_j^*(q_j)\}$ is guaranteed to be a Beta distribution as well. Using a block coordinate descent method that alternatively optimizes $\{\mu_i(z_i)\}$ and $\{\nu_j(q_j)\}$, the mean field (MF) update is

$$\textit{Updating } \mu_i\colon \quad \mu_i(z_i) \propto \prod_{j\in\mathcal{M}_i} a_j^{\delta_{ij}} b_j^{1-\delta_{ij}}, \tag{9}$$

$$\textit{Updating } \nu_j\colon \quad \nu_j(q_j) \sim \mathrm{Beta}\Big(\sum_{i\in\mathcal{N}_j} \mu_i(L_{ij}) + \alpha, \sum_{i\in\mathcal{N}_j} \mu_i(-L_{ij}) + \beta\Big), \tag{10}$$

where $a_j = \exp(\mathbb{E}_{\nu_j}[\ln q_j])$ and $b_j = \exp(\mathbb{E}_{\nu_j}[\ln(1-q_j)])$. The $a_j$ and $b_j$ can be exactly calculated by noting that $\mathbb{E}[\ln x] = \mathrm{Digamma}(\alpha) - \mathrm{Digamma}(\alpha+\beta)$ if $x \sim \mathrm{Beta}(\alpha,\beta)$. One can also instead calculate the first-order approximation of $a_j$ and $b_j$: by Taylor expansion, one have $\ln(1+x) \approx x$; taking $x = (q_j - \bar{q}_j)/\bar{q}_j$, where $\bar{q}_j = \mathbb{E}_{\nu_j}[q_j]$, and substituting it into the definition of $a_j$ and $b_j$, one get $a_j \approx \bar{q}_j$ and $b_j \approx 1 - \bar{q}_j$; it gives an approximate MF (AMF) update,

$$\text{Updating } \mu_i\colon \quad \mu_i(z_i) \propto \prod_{j\in\mathcal{M}_i} \bar{q}_j^{\delta_{ij}}(1-\bar{q}_j)^{1-\delta_{ij}}, \quad \text{Updating } \nu_j\colon \quad \bar{q}_j = \frac{\sum_{i\in\mathcal{N}_j} \mu_i(L_{ij}) + \alpha}{|\mathcal{N}_j| + \alpha + \beta}. \tag{11}$$

The update (11) differs from EM (1) only in replacing $\alpha-1$ and $\beta-1$ with $\alpha$ and $\beta$, corresponding to replacing the posterior mode of the Beta distribution with its posterior mean. This simple (perhaps trivial) difference plays a role of *Laplacian smoothing*, and provides insights for improving the performance of EM. For example, note that the $\hat{q}_j$ in the M-step of EM could be updated to 0 or 1 if $\alpha = 1$ or $\beta = 1$, and once this happens, the $\hat{q}_j$ is locked at its current value, causing EM to trapped at a local maximum. Update (11) can prevent $\bar{q}_j$ from becoming 0 or 1, avoiding the degenerate case. One can of course interpret (11) as EM with prior parameters $\alpha' = \alpha + 1$, and $\beta' = \beta + 1$; under this interpretation, it is advisable to choose priors $\alpha' > 1$ and $\beta' > 1$ (corresponding to a less common or intuitive "informative" prior).

We should point out that it is widely known that EM can be interpreted as a coordinate descent on variational objectives [18, 11]; our derivation differs in that we marginalize, instead of maximize, over $q_j$. Our first-order approximation scheme is also similar to the method by Asuncion [19]. One can also extend this derivation to two-coin models with independent Beta priors, yielding the EM update in Dawid and Skene [2]. On the other hand, discrete priors do not seem to lead to interesting algorithms in this case.

## 4 Experiments

In this section, we present numerical experiments on both simulated and real-world Amazon Mechanical Turk datasets. We implement majority voting (MV), KOS in (2), BP in (8), EM in (1) and

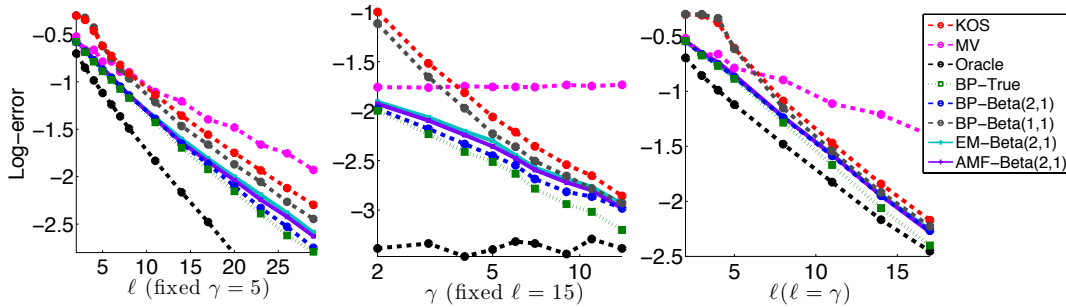

Figure 1: The performance of the algorithms as the degrees of the assignment graph vary; the left degree $\ell$ denotes the number of workers per task, and the right degree $\gamma$ denotes the number of tasks per worker. The true data prior is $\mathrm{prob}[q_j = 0.5] = \mathrm{prob}[q_j = 0.9] = 1/2$.

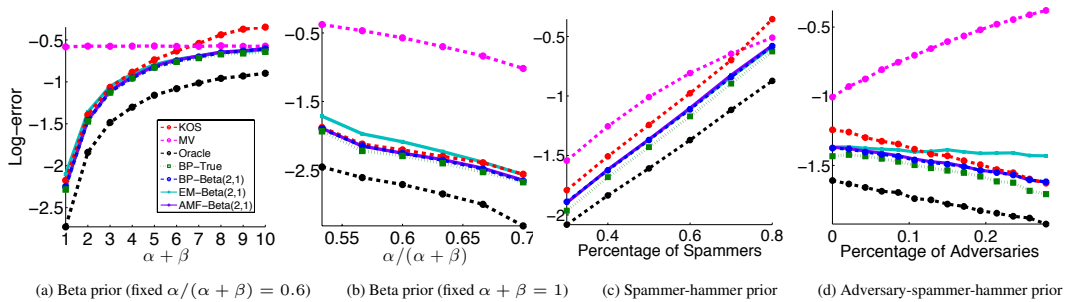

(a) Beta prior (fixed $\alpha/(\alpha + \beta) = 0.6$)  (b) Beta prior (fixed $\alpha + \beta = 1$)  (c) Spammer-hammer prior  (d) Adversary-spammer-hammer prior

Figure 2: The performance on data generated with different $q_j$ priors on (9,9)-regular random graphs. (a) Beta prior with fixed $\frac{\alpha}{\alpha+\beta} = 0.6$. (b) Beta prior with fixed $\alpha + \beta = 1$. (c) Spammer-hammer prior, $\mathrm{prob}[q_j = 0.5] = 1 - \mathrm{prob}[q_j = 0.9] = p_0$, with varying $p_0$. (d) Adversary-spammer-hammer prior, $\mathrm{prob}[q_j = 0.1] = p_0$, $\mathrm{prob}[q_j = 0.5] = \mathrm{prob}[q_j = 0.9] = (1 - p_0)/2$ with varying $p_0$.

its variant AMF in (11). The exact MF (9)-(10) was implemented, but is not reported because its performance is mostly similar to AMF (11) in terms of error rates. We initialize BP (including KOS) with $y_{j \to i} = 1$ and EM with $\mu_i(z_i) = \sum_{j \in \mathcal{M}_i} \mathbb{I}[L_{ij} = z_i]/|\mathcal{M}_i|$, both of which reduce to majority voting at the 0-th iteration; for KOS, we also implemented another version that exactly follows the setting of Karger et al. [1], which initializes $y_{j \to i}$ by $\mathrm{Normal}(1, 1)$ and terminates at the 10-th iteration; the best performance of the two versions was reported. For EM with algorithmic prior $\mathrm{Beta}(\alpha, \beta)$, we add a small constant (0.001) on $\alpha$ and $\beta$ to avoid possible numerical NaN values. We also implemented a max-product version of BP, but found it performed similarly to sum-product BP in terms of error rates. We terminate all the iterative algorithms at a maximum of 100 iterations or with $10^{-6}$ message convergence tolerance. All results are averaged on 100 random trials.

**Simulated Data.** We generate simulated data by drawing the abilities $q_j$ from Beta priors or the *adversary-spammer-hammer* priors, that equals 0.1, 0.5, or 0.9 with certain probabilities; the assignment graphs are randomly drawn from the set of $(\ell, \gamma)$-regular bipartite graphs with 1000 task nodes using the configuration method [20]. For the simulated datasets, we also calculated the oracle lower bound in Karger et al. [1] that assumes the true $q_j$ are *known*, as well as a BP equipped with an algorithmic prior equal to the true prior used to generate the data, which sets a tighter (perhaps approximate) "Bayesian oracle" lower bound for all the algorithms that *do not know* $q_j$. We find that BP and AMF with a typical asymmetric prior $\mathrm{Beta}(2, 1)$ perform mostly as well as the "Bayesian oracle" bound, eliminating the necessity to search for more accurate algorithmic priors.

In Fig. 1, we show that the error rates of the algorithms generally decay exponentially w.r.t. the degree $\ell$ and $\log(\gamma)$ of the assignment graph on a spammer-hammer model. Perhaps surprisingly, we find that the BP, EM and AMF with the asymmetric algorithmic prior $\mathrm{beta}(2, 1)$ scale similarly to KOS, which has been theoretically shown to be order-optimal compared to the oracle lower bound. On the other hand, BP with symmetric algorithmic priors, such as the Haldane prior $\mathrm{Beta}(0^+, 0^+)$ of KOS and the uniform prior $\mathrm{Beta}(1, 1)$, often result in degraded performance when the degrees of the

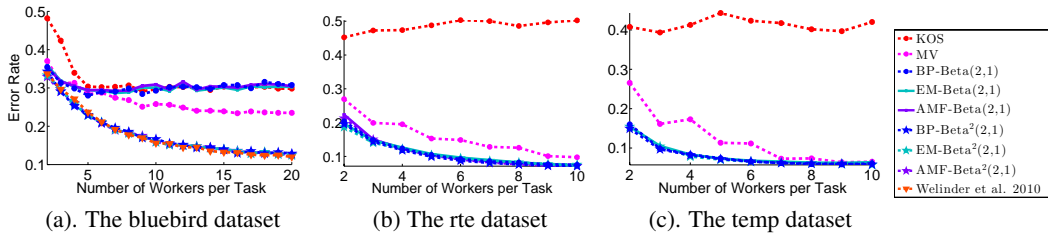

| | | |
|---|---|---|
| (a). The bluebird dataset | (b) The rte dataset | (c). The temp dataset |

Figure 3: The results on Amazon Mechanical Turk datasets. Averaged on 100 random subsamples.

assignment graphs are low, supporting our discussion in Section 3.1.1. Indeed, BP with symmetric algorithmic priors often fails to converge in the low-degree setting.

Fig. 2 shows the performance of the algorithms when varying the true priors of the data. We find in Fig. 2(b) and (d) that the performance of EM with $\text{Beta}(2, 1)$ tends to degrade when the fraction of adversaries increases, probably because the $\hat{q}_j$ is more likely to be incorrectly updated to and stuck on 0 or 1 in these cases; see the discussion in Section 3.2. In all cases, we find that BP and AMF (and MF) perform mostly equally well, perhaps indicating both Bethe and mean-field approximations are reasonably good on the joint distribution $p(z, q|L, \theta)$ in terms of error rates. Our implementation of EM (on both simulated data and the real data below) seems to perform better than previously reported results, probably due to our careful choice on the prior and initialization.

**Real Data.** We tested our methods on three publicly available Amazon Mechanical Turk datasets. The symmetric assumption of $q_j = s_j = t_j$ is likely to be violated in practice, especially on vision datasets where a human's perception decides on whether some object is present. Therefore we also implemented the two-coin version of BP and AMF(EM) with the algorithmic priors of $s_j$ and $t_j$ taken as two independent $\text{Beta}(2, 1)$ (referred to as `BP-Beta`$^2$`(2,1)` and similar).

We first tested on the bluebird dataset of Welinder et al. [6], including 108 tasks and 39 workers on a fully connected bipartite assignment graph, where the workers are asked whether the presented images contain Indigo Bunting or Blue GrosBeak. Fig. 3(a) shows the performance when fixed numbers of annotators are subsampled for each task. On this dataset, all methods, including KOS, BP and AMF(EM), work poorly under the symmetric assumption, while the two-coin versions of BP and AMF(EM) are significantly better, achieving equivalent performance to the algorithm by Welinder et al. [6] based on an advanced high dimensional model. This suggests that the symmetric assumption is badly violated on this dataset, probably caused by the non-expert workers with high sensitivities but low specificities, having trouble identifying Indigo Bunting from Blue GrosBeak.

We then tested on two natural language processing datasets in [21], the rte dataset with 800 tasks and 164 workers, and the temp dataset with 462 tasks and 76 workers. As seen in Fig. 3(b)-(c), both the symmetric and the two-coin versions of BP and AMF(EM) performed equally well, all achieving almost the same performance as the SpEM algorithm reported in [4]. The KOS algorithm does surprisingly poorly, probably due to the assignment graphs not having locally tree-like structures.

## 5   Conclusion

We have presented a spectrum of inference algorithms, in particular a novel and efficient BP algorithm, for crowdsourcing problems and clarified their connections to existing methods. Our exploration provides new insights into the existing KOS, MV and EM algorithms, and more importantly, for separating the *modeling* factors and *algorithmic* factors in crowdsourcing problems, which provides guidance for both implementations of the current algorithms, and for designing even more efficient algorithms in the future. Numerical experiments show that BP, EM and AMF, and exact MF, when implemented carefully, all perform impressively in term of their error rate scaling. Further directions include applying our methodology to more advanced models, e.g., incorporating variation in task difficulties, and theoretical analysis of the error rates of BP, EM and MF that matches the empirical behavior in Fig. 1.

**Acknowledgements**.  Work supported in part by NSF IIS-1065618 and two Microsoft Research Fellowships. We thank P. Welinder and S. Belongie for providing the data and code.

# References

[1] D.R. Karger, S. Oh, and D. Shah. Iterative learning for reliable crowdsourcing systems. In *Neural Information Processing Systems (NIPS)*, 2011.

[2] A.P. Dawid and A.M. Skene. Maximum likelihood estimation of observer error-rates using the em algorithm. *Applied Statistics*, pages 20–28, 1979.

[3] P. Smyth, U. Fayyad, M. Burl, P. Perona, and P. Baldi. Inferring ground truth from subjective labelling of venus images. *Advances in neural information processing systems*, pages 1085–1092, 1995.

[4] V.C. Raykar, S. Yu, L.H. Zhao, G.H. Valadez, C. Florin, L. Bogoni, and L. Moy. Learning from crowds. *The Journal of Machine Learning Research*, 11:1297–1322, 2010.

[5] J Whitehill, P Ruvolo, T Wu, J Bergsma, and J Movellan. Whose vote should count more: Optimal integration of labels from labelers of unknown expertise. In *Advances in Neural Information Processing Systems*, pages 2035–2043. 2009.

[6] P. Welinder, S. Branson, S. Belongie, and P. Perona. The multidimensional wisdom of crowds. In *Neural Information Processing Systems Conference (NIPS)*, 2010.

[7] V.C. Raykar and S. Yu. Eliminating spammers and ranking annotators for crowdsourced labeling tasks. *Journal of Machine Learning Research*, 13:491–518, 2012.

[8] Fabian L. Wauthier and Michael I. Jordan. Bayesian bias mitigation for crowdsourcing. In *Advances in Neural Information Processing Systems 24*, pages 1800–1808. 2011.

[9] B. Carpenter. Multilevel bayesian models of categorical data annotation. *Unpublished manuscript*, 2008.

[10] D. Koller and N. Friedman. *Probabilistic graphical models: principles and techniques*. MIT press, 2009.

[11] M. Wainwright and M. Jordan. Graphical models, exponential families, and variational inference. *Found. Trends Mach. Learn.*, 1(1-2):1–305, 2008.

[12] Y. Weiss and W.T. Freeman. On the optimality of solutions of the max-product belief-propagation algorithm in arbitrary graphs. *Information Theory, IEEE Transactions on*, 47 (2):736 –744, Feb 2001.

[13] F.R. Kschischang, B.J. Frey, and H.A. Loeliger. Factor graphs and the sum-product algorithm. *Information Theory, IEEE Transactions on*, 47(2):498–519, 2001.

[14] D. Tarlow, I.E. Givoni, and R.S. Zemel. Hopmap: Efficient message passing with high order potentials. In *Proc. of AISTATS*, 2010.

[15] A. Zellner. *An introduction to Bayesian inference in econometrics*, volume 17. John Wiley and sons, 1971.

[16] R.E. Kass and L. Wasserman. The selection of prior distributions by formal rules. *Journal of the American Statistical Association*, pages 1343–1370, 1996.

[17] F. Tuyl, R. Gerlach, and K. Mengersen. A comparison of bayes-laplace, jeffreys, and other priors. *The American Statistician*, 62(1):40–44, 2008.

[18] Radford Neal and Geoffrey E. Hinton. A view of the EM algorithm that justifies incremental, sparse, and other variants. In M. Jordan, editor, *Learning in Graphical Models*, pages 355–368. Kluwer, 1998.

[19] A. Asuncion. Approximate mean field for Dirichlet-based models. In *ICML Workshop on Topic Models*, 2010.

[20] B. Bollobás. *Random graphs*, volume 73. Cambridge Univ Pr, 2001.

[21] R. Snow, B. O'Connor, D. Jurafsky, and A.Y. Ng. Cheap and fast—but is it good?: evaluating non-expert annotations for natural language tasks. In *Proceedings of the Conference on Empirical Methods in Natural Language Processing*, pages 254–263. Association for Computational Linguistics, 2008.

